# Outcomes of the Equivalence of Adaptive Ridge with Least Absolute Shrinkage

**Yves Grandvalet**    **Stéphane Canu**

Heudiasyc, UMR CNRS 6599, Université de Technologie de Compiègne,
BP 20.529, 60205 Compiègne cedex, France
Yves.Grandvalet@hds.utc.fr

## Abstract

Adaptive Ridge is a special form of Ridge regression, balancing the quadratic penalization on each parameter of the model. It was shown to be equivalent to Lasso (least absolute shrinkage and selection operator), in the sense that both procedures produce the same estimate. Lasso can thus be viewed as a particular quadratic penalizer.

From this observation, we derive a fixed point algorithm to compute the Lasso solution. The analogy provides also a new hyper-parameter for tuning effectively the model complexity. We finally present a series of possible extensions of lasso performing sparse regression in kernel smoothing, additive modeling and neural net training.

## 1 INTRODUCTION

In supervised learning, we have a set of explicative variables $x$ from which we wish to predict a response variable $y$. To solve this problem, a learning algorithm is used to produce a predictor $\widehat{f}(x)$ from a learning set $s_\ell = \{(x_i, y_i)\}_{i=1}^\ell$ of examples. The goal of prediction may be: 1) to provide an accurate prediction of future responses, accuracy being measured by a user-defined loss function; 2) to quantify the effect of each explicative variable in the response; 3) to better understand the underlying phenomenon.

Penalization is extensively used in learning algorithms. It decreases the predictor variability to improve the prediction accuracy. It is also expected to produce models with few non-zero coefficients if interpretation is planned.

Ridge regression and Subset Selection are the two main penalization procedures. The former is stable, but does not shrink parameters to zero, the latter gives simple models, but is unstable [1]. These observations motivated the search for new penalization techniques such as Garrotte, Non-Negative Garrotte [1], and Lasso (least absolute shrinkage and selection operator) [10].

Adaptive Ridge was proposed as a means to automatically balance penalization on different coefficients. It was shown to be equivalent to Lasso [4]. Section 2 presents Adaptive Ridge and recalls the equivalence statement. The following sections give some of the main outcomes of this connection. They concern algorithmic issues in section 3, complexity control in section 4, and some possible generalizations of lasso to non-linear regression in section 5.

## 2   ADAPTIVE RIDGE REGRESSION

For clarity of exposure, the formulae are given here for linear regression with quadratic loss. The predictor is defined as $\widehat{f}(x) = \beta^T x$, with $\beta^T = (\beta_1, \ldots, \beta_d)$. Adaptive Ridge is a modification of the Ridge estimate, which is defined by the quadratic constraint $\sum_{j=1}^d \beta_j^2 \leq C$ applied to the parameters. It is usually computed by minimizing the Lagrangian

$$\widehat{\beta} = \underset{\beta}{\text{Argmin}} \sum_{i=1}^{\ell} \left( \sum_{j=1}^d \beta_j x_{ij} - y_i \right)^2 + \lambda \sum_{j=1}^d \beta_j^2 \ , \tag{1}$$

where $\lambda$ is the Lagrange multiplier varying with the bound $C$ on the norm of the parameters.

When the ordinary least squares (OLS) estimate maximizes likelihood[1], the Ridge estimate may be seen as a maximum a posteriori estimate. The Bayes prior distribution is a centered normal distribution, with variance proportional to $1/\lambda$. This prior distribution treats all covariates similarly. It is not appropriate when we know that all covariates are not equally relevant.

The garrotte estimate [1] is based on the OLS estimate $\widehat{\beta}^o$. The standard quadratic constraint is replaced by $\sum_{j=1}^d \beta_j^2 / \widehat{\beta}_j^{o^2} \leq C$. The coefficients with smaller OLS estimate are thus more heavily penalized. Other modifications are better explained with the prior distribution viewpoint. Mixtures of Gaussians may be used to cluster different set of covariates. Several models have been proposed, with data dependent clusters [9], or classes defined a priori [7]. The Automatic Relevance Determination model [8] ranks in the latter type. In [4], we propose to use such a mixture, in the form

$$\widehat{\beta} = \underset{\beta}{\text{Argmin}} \sum_{i=1}^{\ell} \left( \sum_{j=1}^d \beta_j x_{ij} - y_i \right)^2 + \sum_{j=1}^d \lambda_j \beta_j^2 \ . \tag{2}$$

Here, each coefficient has its own prior distribution. The priors are centered normal distributions with variances proportional to $1/\lambda_j$. To avoid the simultaneous estimation of these $d$ hyper-parameters by trial, the constraint

$$\frac{1}{d} \sum_{j=1}^d \frac{1}{\lambda_j} = \frac{1}{\lambda} \qquad , \ \lambda_j > 0 \tag{3}$$

is applied on $\boldsymbol{\lambda} = (\lambda_1, \ldots, \lambda_d)^T$, where $\lambda$ is a predefined value. This constraint is a link between the $d$ prior distributions. Their mean variance is proportional to $1/\lambda$. The values of $\lambda_j$ are automatically[2] induced from the sample, hence the qualifier adaptative. Adaptativity refers here to the penalization balance on $\{\widehat{\beta}_j\}$, not to the tuning of the hyper-parameter $\lambda$.

It was shown [4] that Adaptive Ridge and least absolute value shrinkage are equivalent, in the sense that they yield the same estimate. We remind that the Lasso estimate is defined by

$$\widehat{\beta} = \underset{\beta}{\text{Argmin}} \sum_{i=1}^{\ell} \Big( \sum_{j=1}^{d} \beta_j x_{ij} - y_i \Big)^2 \quad \text{subject to} \quad \sum_{j=1}^{d} |\beta_j| \leq K \ . \tag{4}$$

The only difference in the definition of the Adaptive Ridge and the Lasso estimate is that the Lagrangian form of Adaptive Ridge uses the constraint $(\sum_{j=1}^{d} |\beta_j|)^2 / d \leq K^2$.

## 3   OPTIMIZATION ALGORITHM

Tibshirani [10] proposed to use quadratic programming to find the lasso solution, with $2d$ variables (positive and negative parts of $\beta_j$) and $2d + 1$ constraints (signs of positive and negative parts of $\beta_j$ plus constraint (4)). Equations (2) and (3) suggest to use a fixed point (FP) algorithm. At each step $s$, the FP algorithm estimates the optimal parameters $\lambda_j^{(s)}$ of the Bayes prior based on the estimate $\beta_j^{(s-1)}$, and then maximizes the posterior to compute the current estimate $\beta_j^{(s)}$.

As the parameterization $(\beta, \lambda)$ may lead to divergent solutions, we define new variables

$$\gamma_j = \sqrt{\frac{\lambda_j}{\lambda}} \beta_j \ , \quad \text{and} \quad c_j = \sqrt{\frac{\lambda}{\lambda_j}} \quad \text{for } j = 1, \dots, d \ . \tag{5}$$

The FP algorithm updates alternatively $c$ and $\gamma$ as follows:

$$\begin{cases} c_j^{(s)^2} = \dfrac{d \gamma_j^{(s-1)^2}}{\sum_{k=1}^{d} \gamma_k^{(s-1)^2}} \\ \gamma^{(s)} = \big( \text{diag}(c^{(s)}) X^T X \, \text{diag}(c^{(s)}) + \lambda I \big)^{-1} \text{diag}(c^{(s)}) X^T y \ , \end{cases} \tag{6}$$

where $X_{ij} = x_{ij}$, $I$ is the identity matrix, and $\text{diag}(c)$ is the square matrix with the vector $c$ on its diagonal.

The algorithm can be initialized by the Ridge or the OLS estimate. In the latter case, $\beta^{(1)}$ is the garrotte estimate.

Practically, if $\gamma_j^{(s-1)}$ is small compared to numerical accuracy, then $c_j^{(s)}$ is set to zero. In turn, $\gamma_j^{(s)}$ is zero, and the system to be solved in the second step to determine $\gamma$ can be reduced to the other variables. If $c_j$ is set to zero at any time during the optimization process, the final estimate $\widehat{\beta}_j$ will be zero. The computations are simplified, but it is not clear whether global convergence can be obtained with this algorithm. It is easy to show the convergence towards a local minimum, but we did not find general conditions ensuring global convergence. If these conditions exist, they rely on initial conditions.

Finally, we stress that the optimality conditions for $c$ (or in a less rigorous sense for $\lambda$) do not depend on the first part of the cost minimized in (2). In consequence, *the equivalence between Adaptive Ridge and lasso holds for any model or loss function*. The FP algorithm can be applied to these other problems, without modifying the first step.

## 4   COMPLEXITY TUNING

The Adaptive Ridge estimate depends on the learning set $s_\ell$ and on the hyper-parameter $\lambda$. When the estimate is defined by (2) and (3), the analogy with Ridge suggests $\lambda$ as the

"natural" hyper-parameter for tuning the complexity of the regressor. As $\lambda$ goes to zero, $\widehat{\beta}$ approaches the OLS estimate $\widehat{\beta}^{o}$, and the number of effective parameters is $d$. As $\lambda$ goes to infinity, $\widehat{\beta}$ goes to zero and the number of effective parameters is zero.

When the estimate is defined by (4), there is no obvious choice for the hyper-parameter controlling complexity. Tibshirani [10] proposed to use $\nu = \sum_{j=1}^{d} |\widehat{\beta}_j| / \sum_{j=1}^{d} |\widehat{\beta}_j^{o}|$. As $\nu$ goes to one, $\widehat{\beta}$ approaches $\widehat{\beta}^{o}$; as $\nu$ goes to infinity, $\widehat{\beta}$ goes to zero.

The weakness of $\nu$ is that it is explicitly defined from the OLS estimate. As a result, it is variable when the design matrix is badly conditioned. The estimation of $\nu$ is thus harder, and the overall procedure looses in stability. This is illustrated on an experiment following Breiman's benchmark [1] with 30 highly correlated predictors $\mathbb{E}(X_j X_k) = \rho^{|j-k|}$, with $\rho = 1 - 10^{-3}$.

We generate 1000 i.i.d. samples of size $\ell = 60$. For each sample $s_\ell^k$, the modeling error (ME) is computed for several values of $\nu$ and $\lambda$. We select $\nu^k$ and $\lambda^k$ achieving the lowest ME. For one sample, there is a one to one mapping from $\nu$ to $\lambda$. Thus ME is the same for $\nu^k$ and $\lambda^k$. Then, we compute $\nu^*$ and $\lambda^*$ achieving the best average ME on the 1000 samples. As $\nu^k$ and $\lambda^k$ achieve the lowest ME for $s_\ell^k$, the ME for $s_\ell^k$ is higher or equal for $\nu^*$ and $\lambda^*$. Due to the wide spread of $\{\nu_k\}$, the average loss encountered is twice for $\nu^*$ than for $\lambda^*$: $1/1000 \sum_{k=1}^{1000} \left( \mathrm{ME}(s_\ell^k, \nu^*) - \mathrm{ME}(s_\ell^k, \nu^k) \right) = 4.6 \ 10^{-2}$, and $1/1000 \sum_{k=1}^{1000} \left( \mathrm{ME}(s_\ell^k, \lambda^*) - \mathrm{ME}(s_\ell^k, \lambda^k) \right) = 2.3 \ 10^{-2}$. The average modeling error are $\overline{\mathrm{ME}}(\nu^*) = 1.9 \ 10^{-1}$ and $\overline{\mathrm{ME}}(\lambda^*) = 1.7 \ 10^{-1}$.

The estimates of prediction error, such as leave-one-out cross-validation tend to be variable. Hence, complexity tuning is often based on the minimization of some estimate of the mean prediction error (*e.g* bootstrap, K-fold cross-validation). Our experiment supports that, regarding mean prediction error, the optimal $\lambda$ performs better than the optimal $\nu$. Thus, $\lambda$ is the best candidate for complexity tuning.

Although $\lambda$ and $\nu$ are respectively the control parameter of the FP and QP algorithms, the preceding statement does not imply that we should use the FP algorithm. Once the solution $\widehat{\beta}$ is known, $\nu$ or $\lambda$ are easily computed. The choice of one hyper-parameter is not linked to the choice of the optimization algorithm.

## 5 APPLICATIONS

Adaptive Ridge may be applied to a variety of regression techniques. They include kernel smoothing, additive and neural net modeling.

### 5.1 KERNEL SMOOTHING

Soft-thresholding was proved to be efficient in wavelet functional estimation [2]. Kernel smoothers [5] can also benefit from the sparse representation given by soft-thresholding methods. For these regressors, $\widehat{f}(\boldsymbol{x}) = \sum_{i=1}^{\ell} \beta_i K(\boldsymbol{x}, \boldsymbol{x}_i) + \beta_0$, there are as many covariates as pairs in the sample. The quadratic procedure of Lasso with $2\ell + 1$ constraints becomes computationally expensive, but the FP algorithm of Adaptive Ridge is reasonably fast to converge.

An example of least squares fitting is shown in fig. 1 for the motorcycle dataset [5]. On this example, the hyperparameter $\lambda$ has been estimated by .632 bootstrap (with 50 bootstrap replicates) for Ridge and Adaptive Ridge regressions. For tuning $\lambda$, it is not necessary to determine the coefficients $\beta$ with high accuracy. Hence, compared to Ridge regression,

the overall amount of computation required to get the Adaptive Ridge estimate was about six times more important. For evaluation, Adaptive Ridge is ten times faster as Ridge regression as the final fitting uses only a few kernels (11 out of 133).

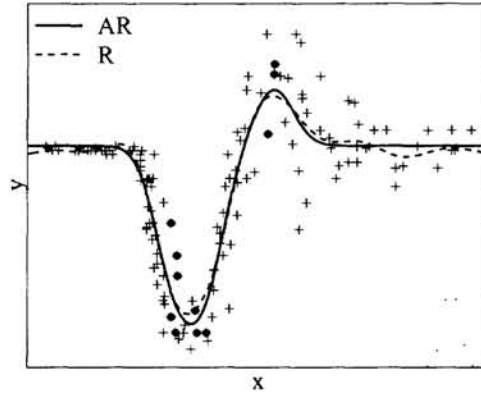

Figure 1: Adaptive Ridge (AR) and Ridge (R) in kernel smoothing on the motorcycle data. The + are data points, and • are the prototypes corresponding to the kernels with non-zero coefficients in AR. The Gaussian kernel used is represented dotted in the lower right-hand corner.

Girosi [3] showed an equivalence between a version of least absolute shrinkage applied to kernel smoothing, and Support Vector Machine (SVM). However, Adaptive Ridge, as applied here, is not equivalent to SVM, as the cost minimized is different. The fit and prototypes are thus different from the fit and support vectors that would be obtained from SVM.

## 5.2 ADDITIVE MODELS

Additive models [6] are sums of univariate functions, $\widehat{f}(x) = \sum_{j=1}^{d} \widehat{f}_j(x_j)$. In the non-parametric setting, $\{\widehat{f}_j\}$ are smooth but unspecified functions. Additive models are easily represented and thus interpretable, but they require the choice of the relevant covariates to be included in the model, and of the smoothness of each $\widehat{f}_j$.

In the form presented in the two previous sections, Adaptive Ridge regression penalizes differently each individual coefficient, but it is easily extended to the pooled penalization of coefficients. Adaptive Ridge may thus be used as an alternative to BRUTO [6] to balance the penalization parameters on each $\widehat{f}_j$.

A classical choice for $\widehat{f}_j$ is cubic spline smoothing. Let $\boldsymbol{B}_j$ denote the $\ell \times (\ell+2)$ matrix of the unconstrained B-spline basis, evaluated at $x_{ij}$. Let $\boldsymbol{\Omega}_j$ be the $(\ell+2) \times (\ell+2)$ matrix corresponding to the penalization of the second derivative of $\widehat{f}_j$. The coefficients of $\widehat{f}_j$ in the unconstrained B-spline basis are noted $\beta_j$. The "natural" extension of Adaptive Ridge is to minimize

$$\left\| \sum_{j=1}^{d} \boldsymbol{B}_j \beta_j - \boldsymbol{y} \right\|^2 + \sum_{j=1}^{d} \lambda_j \beta_j^T \Omega_j \beta_j , \tag{7}$$

subject to constraint (3). This problem is easily shown to have the same solution as the minimization of

$$\left\| \sum_{j=1}^{d} \boldsymbol{B}_j \beta_j - \boldsymbol{y} \right\|^2 + \lambda \left( \sum_{j=1}^{d} \sqrt{\beta_j^T \Omega_j \beta_j} \right)^2 . \tag{8}$$

Note that if the cost (8) is optimized with respect to a single covariate, the solution is a usual smoothing spline regression (with quadratic penalization). In the multidimensional case,

$\alpha_j^2 = \beta_j^T \Omega_j \beta_j = \int \{f_j''(t)\}^2 dt$ may be used to summarize the non-linearity of $f_j$, thus $|\hat{\alpha}_j|$ can be interpreted as a relevance index operating besides linear dependence of feature $j$. The penalizer in (8) is a least absolute shrinkage operator applied to $\alpha_j$. Hence, formula (8) may be interpreted as "quadratic penalization within, and soft-thresholding between covariates".The FP algorithm of section 3 is easily modified to minimize (8), and backfitting may be used to solve the second step of this procedure.

A simulated example in dimension five is shown in fig. 2. The fitted univariate functions are plotted for five values of $\lambda$. There is no dependency between the the explained variable and the last covariate. The other covariates affect the response, but the dependency on the first features is smoother, hence easier to capture and more relevant for the spline smoother. For a small value of $\lambda$, the univariate functions are unsmooth, and the additive model is interpolating the data. For $\lambda = 10^{-4}$, the dependencies are well estimated on all covariates. As $\lambda$ increases, the covariates with higher coordinate number are more heavily penalized, and the corresponding $\hat{f}_j$ tend to be linear.

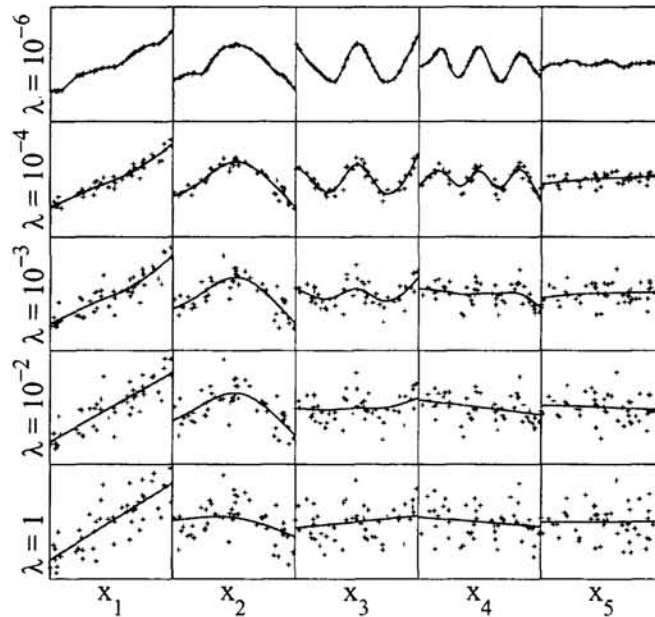

Figure 2: Adaptive Ridge in additive modeling on simulated data. The true model is $y = x_1 + \cos(\pi x_2) + cos(2\pi x_3) + cos(3\pi x_4) + \varepsilon$. The covariates are independently drawn from a uniform distribution on $[-1, 1]$ and $\varepsilon$ is a Gaussian noise of standard deviation $\sigma = 0.3$. The solid curves are the estimated univariate functions for different values of $\lambda$, and $+$ are partial residuals.

Linear trends are not penalized in cubic spline smoothing. Thus, when after convergence $\hat{\beta}_j^T \Omega_j \hat{\beta}_j = 0$, the $j$th covariate is not eliminated. This can be corrected by applying Adaptive Ridge a second time. To test if a significant linear trend can be detected, a linear (penalized) model may be used for $\hat{f}_j$, the remaining $\hat{f}_k$, $k \neq j$ being cubic splines.

## 5.3   MLP FITTING

The generalization to the pooled penalization of coefficients can also be applied to Multi-Layered Perceptrons to control the complexity of the fit. If weights are penalized individually, Adaptive Ridge is equivalent to the Lasso. If weights are pooled by layer, Adaptive Ridge automatically tunes the amount of penalization on each layer, thus avoiding the multiple hyper-parameter tuning necessary in weight-decay [7].

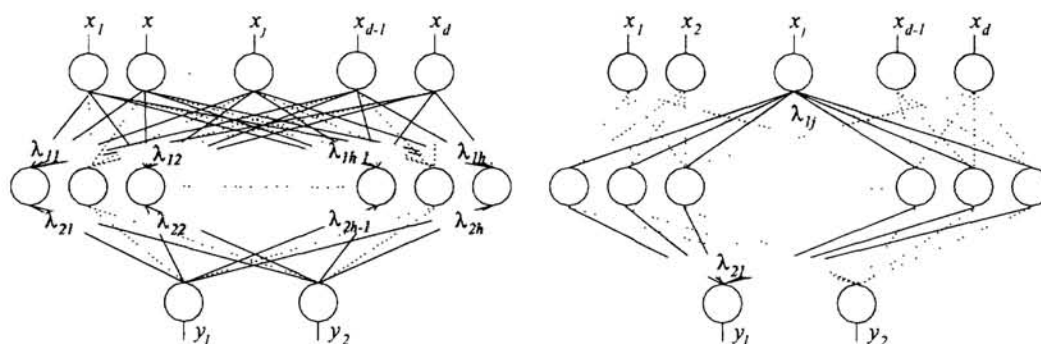

Figure 3: groups of weights for two examples of Adaptive Ridge in MLP fitting. Left: hidden node soft-thresholding. Right: input penalization and selection, and individual smoothing coefficient for each output unit.

Two other interesting configurations are shown in fig. 3. If weights are pooled by incoming and outcoming weights of a unit, node penalization/pruning is performed. The weight groups may also gather the outcoming weights from each input unit, or the incoming weights from each output unit (one set per input plus one per output). The goal here is to penalize/select the input variables according to their relevance, and each output variable according to the smoothness of the corresponding mapping. This configuration proves itself especially useful in time series prediction, where the number of inputs to be fed into the network is not known in advance. There are also more complex choices of pooling, such as the one proposed to encourage additive modeling in Automatic Relevance Determination [8].

## Footnotes

[1] If $\{x_i\}$ are independently and identically drawn from some distribution, and that some $\beta^*$ exists, such that $Y_i = \beta^{*T} x_i + \varepsilon$, where $\varepsilon$ is a centered normal random variable, then the empirical cost based on the quadratic loss is proportional to the log-likelihood of the sample. The OLS estimate $\widehat{\beta}^o$ is thus the maximum likelihood estimate of $\beta^*$.

[2] Adaptive Ridge, as Ridge or Lasso, is not scale invariant, so that the covariates should be normalized to produce sensible estimates.

# References

[1] L. Breiman. Heuristics of instability and stabilization in model selection. *The Annals of Statistics*, 24(6):2350–2383, 1996.

[2] D.L Donoho and I.M. Johnstone. Minimax estimation via wavelet shrinkage. *Ann. Statist.*, 26(3):879–921, 1998.

[3] F. Girosi. An equivalence between sparse approximation and support vector machines. Technical Report 1606, M.I.T. AI Laboratory, Cambridge, MA., 1997.

[4] Y. Grandvalet. Least absolute shrinkage is equivalent to quadratic penalization. In L. Niklasson, M. Bodén, and T. Ziemske, editors, *ICANN'98*, volume 1 of *Perspectives in Neural Computing*, pages 201–206. Springer, 1998.

[5] W. Härdle. *Applied Nonparametric Regression*, volume 19 of *Economic Society Monographs*. Cambridge University Press, New York, 1990.

[6] T.J. Hastie and R.J. Tibshirani. *Generalized Additive Models*, volume 43 of *Monographs on Statistics and Applied Probability*. Chapman & Hall, New York, 1990.

[7] D.J.C. MacKay. A practical Bayesian framework for backprop networks. *Neural Computation*, 4(3):448–472, 1992.

[8] R. M. Neal. *Bayesian Learning for Neural Networks*. Lecture Notes in Statistics. Springer, New York, 1996.

[9] S.J. Nowlan and G.E. Hinton. Simplifying neural networks by soft weight-sharing. *Neural Computation*, 4(4):473–493, 1992.

[10] R.J. Tibshirani. Regression shrinkage and selection via the lasso. *Journal of the Royal Statistical Society, B*, 58(1):267–288, 1995.
